# Empirical performance maximization for linear rank statistics

**Stéphan Clémençon**
Telecom Paristech (TSI) - LTCI UMR Institut Telecom/CNRS 5141
stephan.clemencon@telecom-paristech.fr

**Nicolas Vayatis**
ENS Cachan & UniverSud - CMLA UMR CNRS 8536
vayatis@cmla.ens-cachan.fr

## Abstract

The ROC curve is known to be the golden standard for measuring performance of a test/scoring statistic regarding its capacity of discrimination between two populations in a wide variety of applications, ranging from anomaly detection in signal processing to information retrieval, through medical diagnosis. Most practical performance measures used in scoring applications such as the AUC, the local AUC, the $p$-norm push, the DCG and others, can be seen as summaries of the ROC curve. This paper highlights the fact that many of these empirical criteria can be expressed as (conditional) *linear rank statistics*. We investigate the properties of empirical maximizers of such performance criteria and provide preliminary results for the concentration properties of a novel class of random variables that we will call a *linear rank process*.

## 1 Introduction

In the context of ranking, several performance measures may be considered. Even in the simplest framework of bipartite ranking, where a binary label is available, there is not one and only natural criterion, but many possible options. The ROC curve provides a complete description of performance but its functional nature renders direct optimization strategies rather complex. Empirical risk minimization strategies are thus based on summaries of the ROC curve, which take the form of empirical risk functionals where the averages involved are no longer taken over i.i.d. sequences. The most popular choice is the so-called AUC criterion (see [AGH$^+$05] or [CLV08] for instance), but when top-ranked instances are more important, various choices can be considered: the Discounted Cumulative Gain or DCG [CZ06], the $p$-norm push (see [Rud06]), or the local AUC (refer to [CV07]). The present paper starts from the simple observation that all these summary criteria have a common feature: conditioned upon the labels, they all belong to the class of *linear rank statistics*. Such statistics have been extensively studied in the mathematical statistics literature because of their optimality properties in hypothesis testing, see [HS67]. Now, in the statistical learning view, with the importance of excess risk bounds, the theory of rank tests needs to be revisited and new problems come up. The arguments required to deal with risk functionals based on linear rank statistics have been sketched in [CV07] in a special case. The empirical AUC, known as the Wilcoxon-Mann-Whitney statistic, is also a $U$-statistic and this particular dependence structure was extensively exploited in [CLV08]. In the present paper, we describe the generic structure of linear rank statistics as an orthogonal decomposition after projection onto the space of sums of i.i.d. random variables (Section 2). This projection method is the key to all statistical results related to maximizers of such criteria: consistency, (fast) rates of convergence or model selection. We relate linear rank statistics to performance measures relevant for the ranking problem by showing that the target of ranking algorithms

correspond to optimal ordering rules in that sense (Section 3). Eventually, we provide some preliminary results in Section 4 for empirical maximizers of performance criteria based on linear rank statistics with smooth score-generating functions.

## 2 Criteria based on linear rank statistics

Along the paper, we shall consider the standard binary classification model. Take a random pair $(X, Y) \in \mathcal{X} \times \{-1, +1\}$, where $X$ is an observation vector in a high dimensional space $\mathcal{X} \subset \mathbb{R}^d$ and $Y$ is a binary label, and denote by $P$ the distribution of $(X, Y)$. The dependence structure between $X$ and $Y$ can be described by conditional distributions. We can consider two descriptions: either $P = (\mu, \eta)$ where $\mu$ is the marginal distribution of $X$ and $\eta$ is the posterior distribution defined by $\eta(x) = \mathbb{P}\{Y = 1 \mid X = x\}$ for all $x \in \mathbb{R}^d$, or else $P = (p, G, H)$ with $p = \mathbb{P}\{Y = 1\}$ being the proportion of positive instances, $G = \mathcal{L}(X \mid Y = +1)$ the conditional distribution of positive instances and $H = \mathcal{L}(X \mid Y = -1)$ the conditional distribution of negative instances. A sample of size $n$ of i.i.d. realizations of this statistical model can be represented as a set of pairs $\{(X_i, Y_i)\}_{1 \le i \le n}$, where $(X_i, Y_i)$ is a copy of $(X, Y)$, but also as a set $\{X_1^+, \ldots, X_k^+, X_1^-, \ldots, X_m^-,\}$ where $\mathcal{L}(X_i^+) = G$, $\mathcal{L}(X_i^-) = H$, and $k + m = n$. In this setup, the integers $k$ and $m$ are random, drawn as binomials of size $n$ and respective parameters $p$ and $1 - p$.

### 2.1 Motivation

Most of the statistical learning theory has been developed for empirical risk minimizers (ERM) of sums of i.i.d. random variables. Mathematical results were elaborated with the use of empirical processes techniques and particularly concentration inequalities for such processes (see [BBL05] for an overview). This was made possible by the standard assumption that, in a batch setup, for the usual prediction problems (classification, regression or density estimation), the sample data $\{(X_i, Y_i)\}_{i=1,\ldots,n}$ are i.i.d. random variables. Another reason is that the error probability in these problems involves only "first-order" events, depending only on $(X_1, Y_1)$. In classification, for instance, most theoretical developments were focused on the error probability $\mathbb{P}\{Y_1 \ne g(X_1)\}$ of a classifier $g : \mathcal{X} \to \{-1, +1\}$, which is hardly considered in practice because the two populations are rarely symmetric in terms of proportions or costs. For prediction tasks such as ranking or scoring, more involved statistics need to be considered, such as the Area Under the ROC curve (AUC), the local AUC, the *Discounted Cumulative Gain* (DCG), the $p$-norm push, *etc*. For instance, the AUC, a very popular performance measure in various scoring applications, such as medical diagnosis or credit-risk screening, can be seen as a probability of an "event of order two", *i.e.* depending on $(X_1, Y_1), (X_2, Y_2)$. In information retrieval, the DCG is the reference measure and it seems to have a rather complicated statistical structure. The first theoretical studies either attempt to get back to sums of i.i.d. random variables by artificially reducing the information available (see [AGH+05], [Rud06]) or adopt a plug-in strategy ([CZ06]). Our approach is to i) avoid plug-in in order to understand the intimate nature of the learning problem, ii) keep all the information available and provide the analysis of the full statistic. We shall see that this approach requires the development of new tools for handling the concentration properties of *rank processes*, namely collections of rank statistics indexed by classes of functions, which have never been studied before.

### 2.2 Empirical performance of scoring rules

The learning task on which we focus here is known as the *bipartite ranking problem*. The goal of ranking is to order the instances $X_i$ by means of a real-valued scoring function $s : \mathcal{X} \to \mathbb{R}$, given the binary labels $Y_i$. We denote by $\mathcal{S}$ the set of all scoring functions. It is natural to assume that a good scoring rule $s$ would assign higher ranks to the positive instances (those for which $Y_i = +1$) than to the negative ones. The rank of the observation $X_i$ induced by the scoring function $s$ is expressed as $\mathrm{Rank}(s(X_i)) = \sum_{j=1}^n \mathbb{I}_{\{s(X_j) \le s(X_i)\}}$ and ranges from 1 to $n$. In the present paper, we consider a particular class of simple (conditional) linear rank statistics inspired from the Wilcoxon statistic.

**Definition. 1** *Let* $\phi : [0, 1] \rightarrow [0, 1]$ *be a nondecreasing function. We define the "empirical W-ranking performance measure" as the empirical risk functional*

$$\widehat{W}_n(s) = \sum_{i=1}^n \mathbb{I}_{\{Y_i=+1\}} \phi \left( \frac{\mathrm{Rank}(s(X_i))}{n+1} \right), \ \forall s \in \mathcal{S}.$$

*The function $\phi$ is called the "score-generating function" of the "rank process" $\{\widehat{W}_n(s)\}_{s \in \mathcal{S}}$.*

We refer to the book by Serfling [Ser80] for properties and asymptotic theory of rank statistics. We point out that our definition does not match exactly with the standard definition of linear rank statistics. Indeed, in our case, coefficients of the ranks in the sum are random because they involve the variables $Y_i$. We will call statistics $\widehat{W}_n(s)$ *conditional linear rank statistics*.

It is a very natural idea to consider ranking criteria based on ranks. Observe indeed that the performance of a given scoring function $s$ is invariant by increasing transforms of the latter, when evaluated through the empirical $W$-ranking performance measure. For specific choices of the score-generating function $\phi$, we recover the main examples mentioned in the introduction and many relevant criteria can be accurately approximated by statistics of this form:

- $\phi(u) = u$ - this choice leads to the celebrated Wilcoxon-Mann-Whitney statistic which is related to the empirical version of the AUC (see [CLV08]).

- $\phi(u) = u \cdot \mathbb{I}_{\{u \geq u_0\}}$, for some $u_0 \in (0, 1)$ - such a score-generating function corresponds to the local AUC criterion, introduced recently in [CV07]. Such a criterion is of interest when one wants to focus on the highest ranks.

- $\phi(u) = u^p$ - this is another choice which puts emphasis on high ranks but in a smoother way than the previous one. This is related to the $p$-norm push approach taken in [Rud06]. However, we point out that the criterion studied in the latter work relies on a different definition of the rank of an observation. Namely, the rank of positive instances among negative instances (and not in the pooled sample) is used. This choice permits to use independence which makes the technical part much simpler, at the price of increasing the variance of the criterion.

- $\phi(u) = \phi_n(u) = c\left((n+1)u\right) \cdot \mathbb{I}_{\{u \geq k/(n+1)\}}$ - this corresponds to the DCG criterion in the bipartite setup, one of the "gold standard quality measure" in information retrieval, when *grades* are binary (namely $\mathbb{I}_{\{Y_i=+1\}}$). The $c(i)$'s denote the *discount factors*, $c(i)$ measuring the importance of rank $i$. The integer $k$ denotes the number of top-ranked instances to take into account. Notice that, with our indexation, top positions correspond to the largest ranks and the sequence $\{c_i\}$ should be chosen to be increasing.

## 2.3 Uniform approximation of linear rank statistics

This subsection describes the main result of the present analysis, which shall serve as the essential tool for deriving statistical properties of maximizers of empirical $W$-ranking performance measures. For a given scoring function $s$, we denote by $G_s$, respectively $H_s$, the conditional cumulative distribution function of $s(X)$ given $Y = +1$, respectively $Y = -1$. With these notations, the unconditional cdf of $s(X)$ is then $F_s = pG_s + (1-p)H_s$. For averages of non-i.i.d. random variables, the underlying statistical structure can be revealed by orthogonal projections onto the space of sums of i.i.d. random variables in many situations. This projection argument was the key for the study of empirical AUC maximization, which involved $U$-processes, see [CLV08]. In the case of $U$-statistics, this orthogonal decomposition is known as the *Hoeffding decomposition* and the remainder may be expressed as a degenerate $U$-statistic, see [Hoe48]. For rank statistics, a similar though less accurate decomposition can be considered. We refer to [Haj68] for a systematic use of the *projection method* for investigating the asymptotic properties of general statistics.

**Lemma. 2** *([Haj68]) Let $Z_1, \ldots, Z_n$ be independent r.v.'s and $T = T(Z_1, \ldots, Z_n)$ be a square integrable statistic. The r.v. $\widehat{T} = \sum_{i=1}^n \mathbb{E}[T \mid Z_i] - (n-1)\mathbb{E}[Z]$ is called the Hájek projection of $T$. It satisfies*

$$\mathbb{E}[\widehat{T}] = \mathbb{E}[T] \ and \ \mathbb{E}[(\widehat{T} - T)^2] = \mathbb{E}[(T - \mathbb{E}[T])^2] - \mathbb{E}[(\widehat{T} - \mathbb{E}[\widehat{T}])^2].$$

From the perspective of ERM in statistical learning theory, through the *projection method*, well-known concentration results for standard empirical processes may carry over to more complex collections of r.v. such as *rank processes*, as shown by the next approximation result.

**Proposition. 3** *Consider a score-generating function $\phi$ which is twice continuously differentiable on $[0,1]$. We set $\Phi_s(x) = \phi(F_s(s(x))) + p \int_{s(x)}^{+\infty} \phi'(F_s(u))dG_s(u)$ for all $x \in \mathcal{X}$. Let $\mathcal{S}_0 \subset \mathcal{S}$ be a VC major class of functions. Then, we have: $\forall s \in \mathcal{S}_0$,*

$$\widehat{W}_n(s) = \widehat{V}_n(s) + \widehat{R}_n(s),$$

*where $\widehat{V}_n(s) = \sum_{i=1}^n \mathbb{I}_{\{Y_i=+1\}} \Phi_s(X_i)$ and $\widehat{R}_n(s) = O_{\mathbb{P}}(1)$ as $n \to \infty$ uniformly over $s \in \mathcal{S}$.*

The notation $O_{\mathbb{P}}(1)$ means bounded in probability and the integrals are represented in the sense of the Lebesgue-Stieltjes integral. Details of the proof can be found in the Appendix.

**Remark 1** (ON THE COMPLEXITY ASSUMPTION.) *On the terminology of major sets and major classes, we refer to [Dud99]. In the Proposition 3's proof, we need to control the complexity of subsets of the form $\{x \in \mathcal{X} : s(x) \le t\}$. The stipulated complexity assumption garantees that this collection of sets indexed by $(s,t) \in \mathcal{S}_0 \times \mathbb{R}$ forms a VC class.*

**Remark 2** (ON THE SMOOTHNESS ASSUMPTION.) *We point out that it is also possible to deal with discontinuous score-generating functions as seen in [CV07]. In this case, the lack of smoothness of $\phi$ has to be compensated by smoothness assumptions on the underlying conditional distributions. Another approach would consist of approximating $\widehat{W}_n(s)$ by the empirical W-ranking criterion where the score-generating function $\psi$ would be a smooth approximation of $\phi$. Owing to space limitations, here we only handle the smooth case.*

An essential hint to the study of the asymptotic behavior of a linear rank statistic consists in rewriting it as a function of the sampling cdf. Denoting by $\widehat{F}_s(x) = n^{-1} \sum_{i=1}^n \mathbb{I}_{\{s(X_i) \le x\}}$ the empirical counterpart of $F_s(x)$, we have:

$$\widehat{W}_n(s) = \sum_{i=1}^k \phi\left(\frac{n}{n+1}\widehat{F}_s\left(s(X_i^+)\right)\right).$$

which may easily shown to converge to $\mathbb{E}[\phi(F_s(s(X)) \mid Y = +1]$ as $n \to \infty$, see [CS58].

**Definition. 4** *For a given score-generating function $\phi$, we will call the functional*

$$W_\phi(s) = \mathbb{E}[\phi(F_s(s(X)) \mid Y = +1] ,$$

*a "W-ranking performance measure".*

The following result is a consequence of Proposition 3 and its proof can be found in the Appendix.

**Proposition. 5** *Let $\mathcal{S}_0 \subset \mathcal{S}$ be a VC major class of functions with VC dimension $V$ and $\phi$ be a score-generating function of class $\mathcal{C}^1$. Then, as $n \to \infty$, we have with probability one:*

$$\frac{1}{n} \sup_{s \in \mathcal{S}_0} |\widehat{W}_n(s) - kW_\phi(s)| \to 0.$$

## 3 Optimality

We introduce the class $\mathcal{S}^*$ of scoring functions obtained as strictly increasing transformations of the regression function $\eta$:

$$\mathcal{S}^* = \{ s^* = T \circ \eta \mid T : [0,1] \to \mathbb{R} \text{ strictly increasing } \}.$$

The class $\mathcal{S}^*$ contains the optimal scoring rules for the bipartite ranking problem. The next paragraphs motivate the use of $W$-ranking performance measures as optimization criteria for this problem.

### 3.1 ROC **curves**

A classical tool for measuring the performance of a scoring rule $s$ is the so-called ROC curve

$$\text{ROC}(s,.) : \alpha \in [0,1] \mapsto 1 - G_s \circ H_s^{-1}(1-\alpha),$$

where $H_s^{-1}(x) = \inf\{t \in \mathbb{R} \mid H_s(t) \geq x\}$. In the case where $s = \eta$, we will denote $\text{ROC}(\eta, \alpha) = \text{ROC}^*(\alpha)$, for any $\alpha \in [0,1]$. The set of points $(\alpha, \beta) \in [0,1]^2$ which can be achieved as $(\alpha, \text{ROC}(s, \alpha))$ for some scoring function s is called the ROC space.

It is a well-known fact that the regression function provides an optimal scoring function for the ROC curve. This fact relies on a simple application of Neyman-Pearson's lemma. We refer to [CLV08] for the details. Using the fact that, for a given scoring function, the ROC curve is invariant by increasing transformations of the scoring function, we get the following result:

**Lemma. 6** *For any scoring function $s$ and any $\alpha \in [0,1]$, we have:*

$$\forall s^* \in \mathcal{S}^* , \quad \text{ROC}(s, \alpha) \leq \text{ROC}(s^*, \alpha) \doteq \text{ROC}^*(\alpha) .$$

The next result states that the set of optimal scoring functions coincides with the set of maximizers of the $W_\phi$-ranking performance, provided that the score-generating function $\phi$ is strictly increasing.

**Proposition. 7** *Assume that the score-generating function $\phi$ is strictly increasing. Then, we have:*

$$\forall s \in \mathcal{S} , \quad W_\phi(s) \leq W_\phi(\eta) .$$

*Moreover $W_\phi^* \doteq W_\phi(\eta) = W_\phi(s^*)$ for any $s^* \in \mathcal{S}^*$.*

**Remark 3** (ON PLUG-IN RANKING RULES) *Theoretically, a possible approach to ranking is the* plug-in *method ([DGL96]), which consists of using an estimate $\hat\eta$ of the regression function as a scoring function. As shown by the subsequent bound, when $\phi$ is differentiable with a bounded derivative, when $\hat\eta$ is close to $\eta$ in the $L_1$-sense, it leads to a nearly optimal ordering in terms of W-ranking criterion:*

$$W_\phi^* - W_\phi(\hat\eta) \leq (1-p)||\phi'||_\infty \mathbb{E}[|\hat\eta(X) - \eta(X)|].$$

*However, one faces difficulties with the plug-in approach when dealing with high-dimensional data, see [GKKW02]), which provides the motivation for exploring algorithms based on W-ranking performance maximization.*

### 3.2 **Connection to hypothesis testing**

From the angle embraced in this paper, the ranking problem is tightly related to hypothesis testing. Denote by $X^+$ and $X^-$ two r.v. distributed as $G$ and $H$ respectively. As a first go, we can reformulate the ranking problem as the one of finding a scoring function $s$ such that $s(X^-)$ is *stochastically smaller* than $s(X^+)$, which means, for example, that: $\forall t \in \mathbb{R}, \mathbb{P}\{s(X^-) \geq t\} \leq \mathbb{P}\{s(X^+) \geq t\}$. It is easy to see that the latter statement means that the ROC curve of $s$ dominates the first diagonal of the ROC space. We point out the fact that the first diagonal corresponds to nondiscriminating scoring functions $s_0$ such that $H_{s_0} = G_{s_0}$. However, searching for a scoring function $s$ fulfilling this property is generally not sufficient in practice. Heuristically, one would like to pick an $s$ in order to be as far as possible from the case where "$G_s = H_s$". This requires to specify a certain *measure of dissimilarity* between distributions. In this respect, various criteria may be considered such as the $L^1$-Mallows metric (see the next remark). Indeed, assuming temporarily that $s$ is fixed and considering the problem of testing similarity *vs.* dissimilarity between two distributions $H_s$ and $G_s$ based on two independent samples $s(X_1^+)$, ..., $s(X_k^+)$ and $s(X_1^-)$, ..., $s(X_m^-)$, it is well-known that nonparametric tests based on *linear rank statistics* have optimality properties. We refer to Chapter 9 in [Ser80] for an overview of rank procedures for testing homogeneity, which may yield relevant criteria in the ranking context.

**Remark 4** (CONNECTION BETWEEN AUC AND THE $L^1$-MALLOWS METRIC ) *Consider the* AUC *criterion:* $\text{AUC}(s) = \int_{\alpha=0}^1 \text{ROC}(s, \alpha)d\alpha$. *It is well-known that this criterion may be interpreted as*

the "rate of concording pairs": $\text{AUC}(s) = \mathbb{P}\{s(X) < s(X') \mid Y = -1,\ Y' = +1\}$ where $(X, Y)$ and $(X', Y')$ denote independent copies. Furthermore, it may be easily shown that

$$\text{AUC}(s) = \frac{1}{2} + \int_{-\infty}^{\infty} \{H_s(t) - G_s(t)\} dF(t),$$

where the cdf $F$ may be taken as any linear convex combination of $H_s$ and $G_s$. Provided that $H_s$ is stochastically smaller than $G_s$ and that $F(dt)$ is the uniform distribution over $(0, 1)$ (this is always possible, even if it means replacing $s$ by $F \circ s$, which leaves the ordering untouched), the second term may be identified as the $L^1$-Mallows distance between $H_s$ and $G_s$, a well-known probability metric widely considered in the statistical literature (also known as the $L_1$-Wasserstein metric).

## 4 A generalization error bound

We now provide a bound on the generalization ability of scoring rules based on empirical maximization of W-ranking performance criteria.

**Theorem. 8** *Set the empirical $W$-ranking performance maximizer $\hat{s}_n = \arg\max_{s \in \mathcal{S}} \widehat{W}_n(s)$. Under the same assumptions as in Proposition 3 and assuming in addition that the class of functions $\Phi_s$ induced by $\mathcal{S}_0$ is also a VC major class of functions, we have, for any $\delta > 0$, and with probability $1 - \delta$:*

$$W_\phi^* - W_\phi(\hat{s}_n) \leq c_1 \sqrt{\frac{V}{n}} + c_2 \sqrt{\frac{\log(1/\delta)}{n}},$$

*for some positive constants $c_1$, $c_2$.*

The proof is a straightforward consequence from Proposition 3 and it can be found in the Appendix.

## 5 Conclusion

In this paper, we considered a general class of performance measures for ranking/scoring which can be described as conditional linear rank statistics. Our overall setup encompasses in particular known criteria used in medical diagnosis and information retrieval. We have described the statistical nature of such statistics, proved that they ar compatible with optimal scoring functions in the bipartite setup, and provided a preliminary generalization bound with a $\sqrt{n}$-rate of convergence. By doing so, we provided the very results on a class of linear rank processes. Further work is needed to identify a variance control assumption in order to derive fast rates of convergence and to obtain consistency under weaker complexity assumptions. Moreover, it is not clear how to formulate convex surrogates for such functionals yet.

## Appendix - Proofs

### Proof of Proposition 5

By virtue of the finite increment theorem, we have:

$$\sup_{s \in \mathcal{S}_0} |\widehat{W}_n(s) - k W_\phi(s)| \leq k \|\phi'\|_\infty \left( \frac{1}{n+1} + \sup_{(s,t) \in \mathcal{S}_0 \times \mathbb{R}} |\widehat{F}_s(t) - F_s(t)| \right)$$

and the desired result immediately follows from the application of the VC inequality, see Remark 1.

### Proof of Proposition 3

Since $\phi$ is of class $\mathcal{C}^2$, a Taylor expansion at the second order immediately yields:

$$\widehat{W}_n(s) = \sum_{i=1}^{k} \phi(F_s(s(X_i^+))) + \widehat{B}_n(s) + \widehat{R}_n(s),$$

with

$$\widehat{B}_n(s) = \sum_{i=1}^{k} \left( \frac{\text{Rank}(s(X_i^+))}{n+1} - F_s(s(X_i^+)) \right) \phi'(F_s(s(X_i^+)))$$

$$|\widehat{R}_n(s)| \leq \sum_{i=1}^{k} \left( \frac{\text{Rank}(s(X_i^+))}{n+1} - F_s(s(X_i^+)) \right)^2 ||\phi''||_\infty.$$

Following in the footsteps of [Haj68], we first compute the projection of $\widehat{B}_n(s)$ onto the space $\Sigma$ of r.v.'s of the form $\sum_{i \leq n} f_i(X_i, Y_i)$ such that $\mathbb{E}[f_i^2(X_i, Y_i)] < \infty$ for all $i \in \{1, \ldots, n\}$:

$$P_\Sigma(\widehat{B}_n(s)) = \sum_{i=1}^{k} \sum_{j=1}^{n} \mathbb{E} \left[ \left( \frac{\text{Rank}(s(X_i^+))}{n+1} - F_s(s(X_i^+)) \right) \phi'(F_s(s(X_i))) \mid X_j, Y_j \right],$$

This projection may be splitted into two terms:

$$(I) = \sum_{i=1}^{n} \mathbb{I}_{\{Y_i=+1\}} \left( \frac{1}{n+1} \mathbb{E}[\text{Rank}(s(X_i)) \mid s(X_i)] - F_s(s(X_i)) \right) \phi'(F_s(s(X_i))),$$

$$(II) = \sum_{i=1}^{n} \mathbb{I}_{\{Y_i=+1\}} \sum_{j \neq i} \mathbb{E} \left[ \left( \frac{\text{Rank}(s(X_i^+))}{n+1} - F_s(s(X_i^+)) \right) \phi'(F_s(s(X_i))) \mid s(X_j), Y_j \right].$$

The first term is easily handled and may be seen as negligible (it is of order $O_\mathbb{P}(n^{-1/2})$), since we have $\mathbb{E}[\text{Rank}(s(X_i)) \mid s(X_i)] = n\widehat{F}_s(s(X_i))$ and, by assumption, $\sup_{(s,t) \in \mathcal{S} \times \mathbb{R}} |\widehat{F}_s(t) - F_s(t)| = O_\mathbb{P}(n^{-1/2})$ (see Remark 1). Up to an additive term of order $O_\mathbb{P}(1)$ uniformly over $s \in \mathcal{S}$, the second term may be rewritten as

$$(II) = \frac{1}{n+1} \sum_{i=1}^{n} \mathbb{I}_{\{Y_i=+1\}} \sum_{j \neq i} \mathbb{E} \left[ \mathbb{I}_{\{s(X_j) \leq s(X_i)\}} \phi'(F_s(s(X_i))) \mid s(X_j), Y_j \right]$$

$$= \frac{k}{n+1} \sum_{j=1}^{n} \int_{s(X_j)}^{\infty} \phi'(F_s(u)) dG_s(u) - \frac{1}{n+1} \sum_{i=1}^{n} \mathbb{I}_{\{Y_i=+1\}} \int_{s(X_i)}^{\infty} \phi'(F_s(u)) dG_s(u).$$

As $\sum_{i=1}^{n} \mathbb{I}_{\{Y_i=+1\}} \int_{s(X_i)}^{\infty} \phi'(F_s(u)) dG_s(u)/(n+1) \leq \sup_{u \in [0,1]} \phi'(t)$ and $k/(n+1) \sim p$, we get that, uniformly over $s \in \mathcal{S}_0$:

$$\sum_{i=1}^{k} \phi(F_s(s(X_i^+))) + P_\Sigma(\widehat{B}_n(s))) = \widehat{V}_n(s) + O_\mathbb{P}(1) \text{ as } n \to \infty.$$

The term $\widehat{R}_n(s)$ is negligible, since, up to the multiplicative constant $||\phi''||_\infty$, it is bounded by

$$\frac{1}{(n+1)^2} \sum_{i=1}^{n} \mathbb{E} \left[ \left( 2F_s(s(X_i)) + \sum_{k \neq i} \{\mathbb{I}_{\{s(X_k) \leq s(X_i)\}} - F_s(s(X_i))\} \right)^2 \right].$$

As $F_s$ is bounded by 1, it suffices to observe that for all $i$:

$$\mathbb{E} \left[ \left( \sum_{k \neq i} \{\mathbb{I}_{\{s(X_k) \leq s(X_i)\}} - F_s(s(X_i))\} \right)^2 \mid s(X_i) \right] = \sum_{k \neq i} \mathbb{E} \left[ (\mathbb{I}_{\{s(X_k) \leq s(X_i)\}} - F_s(s(X_i))\})^2 \mid s(X_i) \right].$$

Bounding the variance of the binomial r.v. $\mathbb{E}[\mathbb{I}\{s(X_k) \leq s(X_i)\} - F_s(s(X_i))\}^2 \mid s(X_i)]$ by $1/4$, one finally gets that $\widehat{R}_n(s)$ is of order $O_\mathbb{P}(1)$ uniformly over $s \in \mathcal{S}_0$.

Eventually, one needs to evaluate the accuracy of the approximation yield by the projection $\widehat{B}_n(s) - \{P_\Sigma(\widehat{B}_n(s)) - (n-1)\mathbb{E}[\widehat{B}_n(s)]\}$. Write, for all $s \in \mathcal{S}_0$,

$$\widehat{B}_n(s) = n\widehat{U}_n(s) + \sum_{i=1}^{n} \mathbb{I}_{\{Y_i=+1\}} \left( \frac{1}{n+1} - F_s(s(X_i)) \right) \phi'(F_s(s(X_i))),$$

where $U_n(s) = \sum_{i \neq j} q_s((X_i, Y_i), (X_j, Y_j))/(n(n+1))$ is a $U$-statistic with kernel:

$$q_s((x, y), (x', y')) = \mathbb{I}_{\{y=+1\}} \cdot \mathbb{I}_{\{s(x') \leq s(x)\}} \cdot \phi'(F_s(s(x))).$$

Hence, we have

$$n^{-1}\left(\widehat{B}_n(s) - \{P_\Sigma(\widehat{B}_n(s)) - (n-1)\mathbb{E}[\widehat{B}_n(s)]\}\right) = \widehat{U}_n(s) - \{P_\Sigma(\widehat{U}_n(s)) - (n-1)\mathbb{E}[\widehat{U}_n(s)]\}$$

which actually corresponds to the degenerate part of the *Hoeffding decomposition* of the $U$-statistic $\widehat{U}_n(s)$. Now, given that $\sup_{s \in \mathcal{S}_0} ||q_s||_\infty < \infty$, it follows from Theorem 11 in [CLV08] for instance, combined with the basic symmetrization device of the kernel $q_s$, that

$$\sup_{s \in \mathcal{S}_0} |\widehat{U}_n(s) - \{P_\Sigma(\widehat{U}_n(s))\mathbb{E}[\widehat{U}_n(s)]\}| = O_\mathbb{P}(n^{-1}) \text{ as } n \to \infty,$$

which concludes the proof.

**Proof of Proposition 7**

Using the decomposition $F_s = pG_s + (1-p)H_s$, we are led to the following expression:

$$pW_\phi(s) = \int_0^1 \phi(u) \, du - (1-p)\mathbb{E}[\phi(F_s(s(X))) \mid Y = -1].$$

Then, using a change of variable:

$$\mathbb{E}[\phi(F_s(s(X))) \mid Y = -1] = \int_0^1 \phi(p(1 - \text{ROC}(s, \alpha)) + (1-p)(1-\alpha)) \, d\alpha \,.$$

It is now easy to conclude since $\phi$ is increasing (by assumption) and because of the optimality of elements of $\mathcal{S}^*$ in the sense of Lemma 6.

**Proof of Theorem 8**

Observe that, by virtue of Proposition 3,

$$W_\phi^* - W_\phi(\hat{s}_n) \leq 2 \sup_{s \in \mathcal{S}_0} |\widehat{W}_n(s)/k - W_\phi(s)| \leq \frac{2}{k} \sup_{s \in \mathcal{S}_0} |\widehat{V}_n(s) - kW_\phi(s)| + O_\mathbb{P}(n^{-1}),$$

and the desired bound derives from the VC inequality applied to the sup term, noticing that it follows from our assumptions that $\{(x, y) \mapsto \mathbb{I}_{\{y=+1\}}\Phi_s(x)\}_{s \in \mathcal{S}_0}$ is a *VC* class of functions.

## References

[AGH+05] S. Agarwal, T. Graepel, R. Herbrich, S. Har-Peled, and D. Roth. Generalization bounds for the area under the ROC curve. *Journal of Machine Learning Research*, 6:393–425, 2005.

[BBL05] S. Boucheron, O. Bousquet, and G. Lugosi. Theory of Classification: A Survey of Some Recent Advances. *ESAIM: Probability and Statistics*, 9:323–375, 2005.

[CLV08] S. Clémençon, G. Lugosi, and N. Vayatis. Ranking and empirical risk minimization of U-statistics. *The Annals of Statistics*, 36(2):844–874, 2008.

[CS58] J. Chernoff and Savage. Asymptotic normality and efficiency of certain non parametric test statistics. *Ann. Math. Stat.*, 29:972–994, 1958.

[CV07] S. Clémençon and N. Vayatis. Ranking the best instances. *Journal of Machine Learning Research*, 8:2671–2699, 2007.

[CZ06] D. Cossock and T. Zhang. Subset ranking using regression. In H.U. Simon and G. Lugosi, editors, *Proceedings of COLT 2006*, volume 4005 of *Lecture Notes in Computer Science*, pages 605–619, 2006.

[DGL96] L. Devroye, L. Györfi, and G. Lugosi. *A Probabilistic Theory of Pattern Recognition*. Springer, 1996.

[Dud99]     R.M. Dudley. *Uniform Central Limit Theorems*. Cambridge University Press, 1999.

[GKKW02] L. Györfi, M. Köhler, A. Krzyzak, and H. Walk. *A Distribution-Free Theory of Non-parametric Regression*. Springer, 2002.

[Haj68]      J. Hajek. Asymptotic normality of simple linear rank statistics under alternatives. *Ann. Math. Stat.*, 39:325–346, 1968.

[Hoe48]     W. Hoeffding. A class of statistics with asymptotically normal distribution. *Ann. Math. Stat.*, 19:293–325, 1948.

[HS67]       J. Hájek and Z. Sidák. *Theory of Rank Tests*. Academic Press, 1967.

[Rud06]     C. Rudin. Ranking with a P-Norm Push. In H.U. Simon and G. Lugosi, editors, *Proceedings of COLT 2006*, volume 4005 of *Lecture Notes in Computer Science*, pages 589–604, 2006.

[Ser80]      R.J. Serfling. *Approximation theorems of mathematical statistics*. John Wiley & Sons, 1980.

